# Kernel-based Extraction of Slow Features: Complex Cells Learn Disparity and Translation Invariance from Natural Images

**Alistair Bray and Dominique Martinez***
CORTEX Group, LORIA-INRIA, Nancy, France
*bray@loria.fr, dmartine@loria.fr*

## Abstract

In Slow Feature Analysis (SFA [1]), it has been demonstrated that high-order invariant properties can be extracted by projecting inputs into a nonlinear space and computing the slowest changing features in this space; this has been proposed as a simple general model for learning nonlinear invariances in the visual system. However, this method is highly constrained by the curse of dimensionality which limits it to simple theoretical simulations. This paper demonstrates that by using a different but closely-related objective function for extracting slowly varying features ([2, 3]), and then exploiting the kernel trick, this curse can be avoided. Using this new method we show that both the complex cell properties of translation invariance and disparity coding can be learnt simultaneously from natural images when complex cells are driven by simple cells also learnt from the image.

The notion of maximising an objective function based upon the temporal predictability of output has been progressively applied in modelling the development of invariances in the visual system. Földiák used it indirectly via a Hebbian trace rule for modelling the development of translation invariance in complex cells [4] (closely related to many other models [5, 6, 7]); this rule has been used to maximise invariance as one component of a hierarchical system for object and face recognition [8]. On the other hand, similar functions have been maximised directly in networks for extracting linear [2] and nonlinear [9, 1] visual invariances. Direct maximisation of such functions have recently been used to model complex cells [10] and as an alternative to maximising sparseness/independence in modelling simple cells [11].

Slow Feature Analysis [1] combines many of the best properties of these methods to provide a good general nonlinear model. That is, it uses an objective function that minimises the first-order temporal derivative of the outputs; it provides a closed-form solution which maximises this function by projecting inputs into a nonlinear

space; it exploits sphering (or PCA-whitening) of the data to ensure that all outputs have unit variance and are uncorrelated. However, the method suffers from the curse of dimensionality in that the nonlinear feature space soon becomes very large as the input dimension grows, and yet this feature space must be represented explicitly in order for the essential sphering to occur.

The alternative that we propose here is to use the objective function of Stone [2, 9], that maximises output variance over a long period whilst minimising variance over a shorter period; in the linear case, this can be implemented by a biologically plausible mixture of Hebbian and anti-Hebbian learning on the same synapses [2]. In recent work, Stone has proposed a closed-form solution for maximising this function in the linear domain of blind source separation that does not involve data-sphering. This paper describes how this method can be kernelised. The use of the "kernel trick" allows projection of inputs into a nonlinear kernel induced feature space of very high (possibly infinite) dimension which is never explicitly represented or accessed. This leads to an efficient method that maps to an architecture that could be biologically implemented either by Sigma-Pi neurons, or fixed RBF networks (as described for SFA [1]). We demonstrate that using this method to extract features that vary slowly in natural images leads to the development of both the complex-cell properties of translation invariance and disparity coding simultaneously.

## 1 Finding Slow Features with kernels

Given $l$ time-series vectors $\mathbf{x}_{i<l}$ where each $n$-dimensional vector $x_i$ is a linear mixture of $n$ unknown but temporally predictable parameters at time $i$, the problem in [3] is to find an $n$-dimensional weight vector $\mathbf{w}$ so that the output $y_i = \mathbf{w}^T\mathbf{x}_i$ at each $i$ is a scaled version of a particular parameter. Many quasi-invariant parameters underlying perceptual data exhibit these properties of short-term predictability and long-term variability. Accordingly, an objective function F can be defined as the ratio between the long-term variance V and the short-term variance S of the output sequence i.e.

$$F = \frac{V}{S} = \frac{\sum_i \overline{y_i}^2}{\sum_i \widetilde{y_i}^2} \tag{1}$$

where $\overline{y}_i$ and $\widetilde{y}_i$ represent the output at $i$ centered using long- and short-term means. The aim is to find the parameters that maximize $F$, which can be rewritten as:

$$F = \frac{\mathbf{w}^T\overline{\mathbf{C}}\mathbf{w}}{\mathbf{w}^T\widetilde{\mathbf{C}}\mathbf{w}} \text{ where } \overline{\mathbf{C}} = \frac{1}{l}\sum_i \overline{\mathbf{x}}_i\overline{\mathbf{x}}_i^T \text{ and } \widetilde{\mathbf{C}} = \frac{1}{l}\sum_i \widetilde{\mathbf{x}}_i\widetilde{\mathbf{x}}_i^T$$

where $\overline{\mathbf{C}}$ and $\widetilde{\mathbf{C}}$ are $nxn$ covariance matrices estimated from the $l$ inputs. $F$ is a version of the Rayleigh quotient and the problem to be solved is, in analogy to PCA, the right-handed generalized symmetric eigenproblem:

$$\overline{C}\mathbf{w} = \lambda\widetilde{C}\mathbf{w} \tag{2}$$

where $\lambda$ is the largest eigenvalue and $\mathbf{w}$ the corresponding eigenvector. In this case, the component extracted $y = \mathbf{w}^T \mathbf{x}$ corresponds to the most predictable component with $F = \lambda$. Most importantly, more than one component can be extracted by considering successive eigenvalues and eigenvectors which are orthogonal in the metrics $\overline{C}$ and $\widetilde{C}$, i.e. $\mathbf{w}_i^T \overline{\mathbf{C}} \mathbf{w}_j = 0$ and $\mathbf{w}_i^T \widetilde{\mathbf{C}} \mathbf{w}_j = 0$ for $i \neq j$.

To make this algorithm nonlinear we can first project the data $\mathbf{x}$ into some high-dimensional feature space via a nonlinear mapping $\phi$, and then find the weight vector $\mathbf{w}$ that maximizes $F$ in this space. In this case, to optimise Eq. (2) the covariance matrices must be estimated in the feature space as

$$\overline{C} = \frac{1}{l} \sum_i \overline{\phi(\mathbf{x}_i)} \, \overline{\phi(\mathbf{x}_i)}^T \text{ and } \widetilde{C} = \frac{1}{l} \sum_i \widetilde{\phi(\mathbf{x}_i)} \, \widetilde{\phi(\mathbf{x}_i)}^T$$

where $\overline{\phi(\mathbf{x}_i)}$ and $\widetilde{\phi(\mathbf{x}_i)}$ represent the data centered in the feature space. The problem with this straight-forward approach is that the dimensionality of the feature space quickly becomes huge as the input dimension increases [1]. To prevent this we use the kernel trick: to avoid working with the mapped data directly, we assume that the solution $\mathbf{w}$ can be written as an expansion in terms of mapped training data: $\mathbf{w} = \sum_{i=1}^{l} \alpha_i \phi(\mathbf{x}_i)$. We can now rewrite the numerator (likewise denominator) in $F$ as

$$
\begin{aligned}
\mathbf{w}^T \overline{C} \mathbf{w} &= \frac{1}{l} \sum_{i,j} \alpha_i \alpha_j \phi(\mathbf{x}_i)^T \sum_k \overline{\phi(\mathbf{x}_k)} \, \overline{\phi(\mathbf{x}_k)}^T \phi(\mathbf{x}_j) \\
&= \frac{1}{l} \sum_{i,j} \alpha_i \alpha_j \sum_k \phi(\mathbf{x}_i)^T \overline{\phi(\mathbf{x}_k)} \phi(\mathbf{x}_j)^T \overline{\phi(\mathbf{x}_k)} \\
&= \frac{1}{l} \alpha^T \overline{\mathbf{K}} \, \overline{\mathbf{K}}^T \alpha
\end{aligned}
$$

where $\alpha = (\alpha_1 \cdots \alpha_l)^T$ and $\overline{\mathbf{K}}$ is a $(l x l)$ matrix with entries defined as $\overline{K}_{ij} = \phi(\mathbf{x}_i)^T \overline{\phi(\mathbf{x}_j)}$. F can now be written as:

$$F = \frac{\alpha^T \overline{\mathbf{K}} \, \overline{\mathbf{K}}^T \alpha}{\alpha^T \widetilde{\mathbf{K}} \, \widetilde{\mathbf{K}}^T \alpha} \tag{3}$$

To avoid explicitly computing dot products in the feature space, we introduce kernel functions defined as $k(\mathbf{x}, \mathbf{y}) = \phi(\mathbf{x})^T \phi(\mathbf{y})$, which means we just have to evaluate kernels in the input space. Any kernel involved in Support Vector Machines can be used, e.g. linear, polynomial, RBF or sigmoid. By now defining the kernel matrix $\mathbf{K}$ with entries

$$\mathbf{K}_{ij} = k(\mathbf{x}_i, \mathbf{x}_j) = \phi(\mathbf{x}_i)^T \phi(\mathbf{x}_j) \tag{4}$$

we can arrive at the corresponding eigenproblem:

$$\overline{\mathbf{K}}\,\overline{\mathbf{K}}^T \alpha = \lambda\,\widetilde{\mathbf{K}}\,\widetilde{\mathbf{K}}^T \alpha \tag{5}$$

where $\lambda$ is again the corresponding largest eigenvalue equal to F. As for the linear case, more than one source can be extracted by considering successive eigenvalues and eigenvectors. In order to recover a temporal component, we need only to compute the nonlinear projection $y = \mathbf{w}^T \phi(\mathbf{x})$ of a new input $\mathbf{x}$ onto $\mathbf{w}$ which is equivalent to $y = \sum_{i=1}^{l} \alpha_i k(\mathbf{x}_i, \mathbf{x})$.

**Finding a sparse solution**

If the eigen problem is solved on the entire training set then this algorithm also suffers from the curse of dimensionality, since the matrices ($lxl$) easily become computationally intractable. A sparse solution using a small subset $p$ of the training data in the expansion is therefore essential: this is called the basis set BS. The output is now $y = \sum_{i \in BS} \alpha_i k(\mathbf{x}_i, \mathbf{x})$, and the solution must lie in the subspace spanned by BS. The kernel elements $\mathbf{K}_{ij}$ are computed between the $p$ basis vectors $\mathbf{x}_i$ and the $l$ training data $\mathbf{x}_j$. Thus, $\mathbf{K}$, $\overline{\mathbf{K}}$ and $\widetilde{\mathbf{K}}$ are rectangular $pxl$ but the covariance matrices ($\overline{\mathbf{K}}\,\overline{\mathbf{K}}^T$) and ($\widetilde{\mathbf{K}}\,\widetilde{\mathbf{K}}^T$) used in the eigenproblem are only $pxp$. This approach can effectively solve very large problems, provided $p << l$. The question of course is how to choose the basis vectors: it is both necessary and sufficient that they span the space of the solution in the kernel induced feature space. In a recent version of the algorithm [12] we use the sparse greedy method of [13] as a preprocessing step. This efficiently finds a small basis set that minimises the least-squares error between data points in feature space and those reconstructed in the feature space defined by the basis set. In the simulations below we used a less efficient greedy algorithm that performed equally well here, but requires a considerably larger basis set[1].

The complete online algorithm requires minimal memory, making it ideal for very large data sets. The implementation estimates the long- and short-term kernel means online using exponential time averages parameterised using half-lives $\Lambda_s, \Lambda_l$ (as in [9]). Likewise, the covariance matrices $\overline{\mathbf{K}\mathbf{K}}^T$, $\tilde{\mathbf{K}}\tilde{\mathbf{K}}^T$ are updated online at each time step e.g. $\overline{\mathbf{K}\mathbf{K}}^T$ is updated to $\overline{\mathbf{K}\mathbf{K}}^T + \overline{\mathcal{K}\mathcal{K}}^T$ where $\mathcal{K}$ is the column vector of kernel values centred using the long term mean and computed for the current time step; there is therefore no need to explicitly compute or store kernel matrices.

## 2 Simulation Results

The simulation was performed using a grey-level stereo pair of resolution 128x128, shown in Figure 1[a]. A new 2D direction $0° < \theta \le 360°$ was selected at every 64 time steps, and the image was translated by one pixel per time step in this direction (with toroidal wrap-around).

A set of 20 monocular simple cells was learnt using the algorithm described in [11] that maximises a nonlinear measure of temporal correlation (TRS) between the

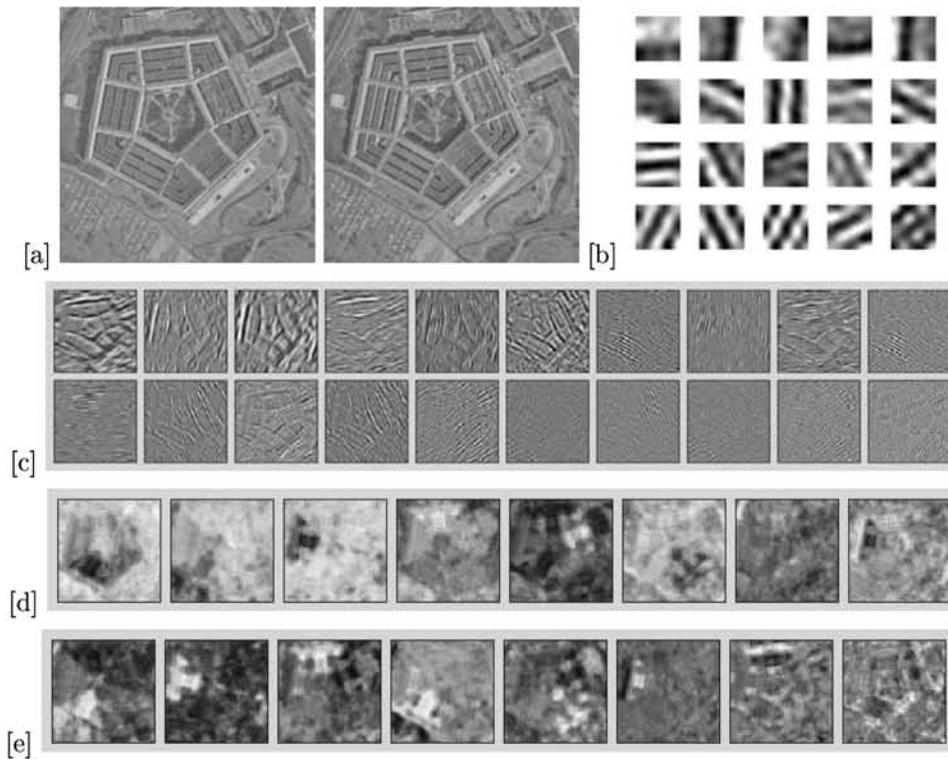

Figure 1: Training on natural images. [a] Stereo Pair. [b] Linear filters that maximise TRS [11]. [c] Output of filters for left image. [d] Output of nonlinear complex cells in binocular simulation. [e] Output of complex cells in monocular simulation.

present and a previous output, based upon the transfer function $g(y) = \ln \cosh(y)$. We chose this algorithm since it is based on a nonlinear measure of temporal correlation and yet provides a linear sparse-distributed coding, very similar to that of ICA for describing simple cells [14]. We did not use the objective function described above since in the linear case it yields filters similar to the local Fourier series[2]. The filters were optimised for this particular stereo pair; simulations using a greater variation of more natural images resulted in more spatially localised filters very similar to those in [14, 11]. We used only the 20 most predictable filters since results did not improve through use of the full set. The simple cell receptive field was 8x8, and during learning data was provided by both eyes at one position in the image[3]. The oriented Gabor-like weight vectors for the 20 cells contributing most to the TRS objective function are shown in Figure 1[b], and the result of processing the left image with these linear filters is shown in Figure 1[c].

The complex cells received input from these 20 types of simple cells when processing both the left and right eye images. Complex cells had a spatial receptive field of 4x4;

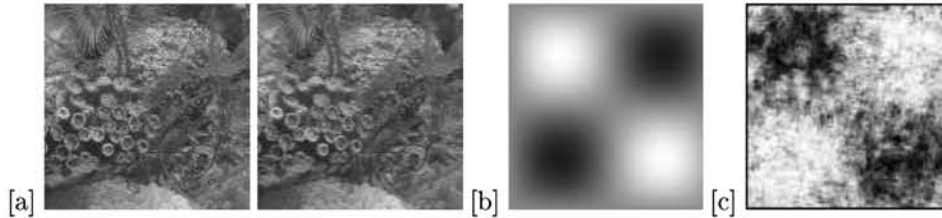

Figure 2: Testing on simulated pair used in [9]. [a] Artificial stereo pair. [b] Underlying disparity function. [c] Output of most predictable complex cell trained on Figure 1[a].

each cell therefore received 320 simple cell inputs (2x4x4x20); these were normalised to have unit variance and zero mean. The most predictable features were extracted for this input vector over $10^5$ time-steps, using the kernel-based method described above, using data at just one position in the image. The basis set was made up of 400 input vectors, and a polynomial kernel of degree 2 was used. The temporal half-lifes for estimating the short- and long-term means in U and V were $\Lambda_s = 2, \Lambda_l = 200$. The algorithm therefore extracts 400 outputs; we display the outputs for the 8 most predictable (determined by highest eigenvalues) in Figure 1[d]; further values were hard to interpret. Below this, in Figure 1[e], we show the complex outputs obtained if we substitute the right image with the left one in the stereo pair, so making the simulation monocular.

Consider first the monocular simulation in [e]. It is visually apparent how the most predictable units are strongly selective for regions of iso-orientation (looking quite different to any simple cell response in [c]). In this particular image, it results in different "**T**"-shaped parts of the Pentagon of considerable size being distinctly isolated. Since in our network the complex cell receptive field size in the image is only 50% greater than that for the simple cells, this implies translation invariance: over the time (or space) that a simple cell of the correct orientation gives a strong but transitory response, the complex cells provides a strong continuous response. That is, its response is invariant to the phase that determines the profile of the simple cell response.

Consider now the stereo simulation in [d]. This tendency is still present (e.g. the 3rd output), but it is confounded with another parameter that isolates the complete shape of the Pentagon from the background. This is most striking in the output provided by the first feature; that is, this parameter is the most predictable in the image (providing an eigenvalue $\lambda = V/U = 7.28$, as opposed to $\lambda \approx 4$ for the "**T**"-shapes in [e]). This parameter is binocular disparity, generated by the variation in depth of the Pentagon roof compared to the ground. The proof of this lies in Figure 2. Here we have taken the artificial stereo pair used in [9], shown in Figure 2[a], that has been generated using the known eggshell disparity function shown in Figure 2[b]. We presented this to the network trained wholly on the Pentagon stereo pair; it can be seen that the most predictable component, shown in Figure 2[c], replicates the disparity function of [b][4].

# 3   Discussion

The simulation above confirms that the linear properties of simple cells, and two of the nonlinear properties of complex cells (translation invariance and disparity coding) can be extracted simultaneously from natural images through maximising functions of temporal coherence in the input. Although these properties have been dealt with in others' work discussed above, they have been considered either in isolation or through theoretical simulation. It is only because the kernel-based method we present allows us to work efficiently with large amounts of data in a nonlinear feature space derived from high dimensional input that we have been able to extract both complex cell properties together from realistic image data.

The method described above is computationally efficient. It is also biologically plausible in as much as [a] it uses a reasonable objective function based on temporal coherence of output, and [b] the final computation required to extract these most predictable outputs could be performed either by Sigma-Pi neurons, or fixed RBF networks (as in SFA [1]). However, we do not claim either that the precise formulation of the objective function is biologically exact, or that a biological system would use the same means to arrive at the final architecture that computes the optimal solution: the learning algorithm is certainly different. Our approach is therefore focussed on the constraints provided by [a] and [b].

The method also exploits a distributed representation for maximising the objective function that results from the generalised eigenvector solution. Is this plausible given the emphasis that has been laid on sparse-coding early in the visual system [15]? Sparse representations are often the result of constraining different outputs to be uncorrelated, or stronger, independent. However, as one ascends the perceptual pathway generating more reduced nonlinear representations, even the constraint of uncorrelated output may be too strong, or unnecessary, to create the highly robust representations exploited by the brain. For example, Rolls reports and defends a highly distributed coding of faces in infero-temporal cortical areas with cells responding to a large proportion of stimuli to some degree ([16], chapter 5). Our method enforces the constraint that successive eigenvectors are orthogonal in the metrics $\overline{C}$ and $\widehat{C}$ and can result in the partly correlated output expected in the robust distributed coding Rolls proposes. However, this would not be the case if the long-term means used for $\overline{C}$ are estimated with a temporal half-life sufficiently large that these means do not differ from the true expected values.

Finally, although maximising the sparseness of representation may be inappropriate in deeper cortex, one might suggest that the coding of parameters we obtain in our simulation is not highly distributed across outputs: in reality each complex cell responds to a limited range of disparity and orientation. However, it can be seen in Figure 1[d]) that there is a clear separation of orientation, and some mixing of disparity and orientation-sensitivity. It is a feature of our method that different outputs must have different measures of predictability (i.e. eigenvalues). In the case of sparse coding of translation invariance, for example, there is no obvious reason why this assumption should be met by cells coding different orientations alone; it can however be enforced by coding different mixtures of orientation and disparity parameters leading to distinct eigenvalues. There is certainly no practical or biological reason why these parameters should be carried separately in the visual system (see [1] for discussion).

In conclusion, this work provides further support for the fruitful approach of extracting non-trivial parameters through maximisation of objective functions based on temporal properties of perceptual input. One of the challenges here is to extend current linear models into the nonlinear domain whilst limiting the extra complexity they bring, which can lead to excess degrees of freedom and computational problems. We have described here a kernel-based method that goes some way towards this, extracting disparity and translation simultaneously for complex cells trained on natural images.

## Footnotes

*http://www.loria.fr/equipes/cortex/

[1]Vectors $x$ are added to BS if, for $y \in BS$, $|k(x,y)| \le \tau$ where threshold $\tau$ is slowly annealed from $\tau_0 = 1$, and the size of BS is set at 400.

[2]An intuitive explanation for this necessity for nonlinearity in the objective function is provided in [11]; in brief, the temporal correlation of the output of a Gabor-like linear filter is low, whilst a similar correlation for a measure of the *power* in the filter is high.

[3]The dimension of the PCA-whitened space was reduced from 63 to 40, and $\Delta t = 1, \eta = 10^{-3}, \alpha = 10^{-1}$; $10^5$ input vectors were used.

[4]The output is somewhat noisy, partly because the image has few linear features like those in Figure 1[b]; if we train the simple and complex cells on this image we get a much cleaner result.

# References

[1] L. Wiskott and T.J. Sejnowski. Slow feature analysis: Unsupervised learning of invariances. *Neural Computation*, 14(4), 2002.

[2] J. V. Stone and A. J. Bray. A learning rule for extracting spatio-temporal invariances. *Network: Computation in Neural Systems*, 6(3):429–436, 1995.

[3] James V. Stone. Blind source separation using temporal predictability. *Neural Computation*, (13):1559–1574, 2001.

[4] P. Földiák. Learning invariance from transformation sequences. *Neural Computation*, 3(2):194–200, 1991.

[5] H. G. Barrow and A. J. Bray. A model of adaptive development of complex cortical cells. In I. Aleksander and J. Taylor, editors, *Artificial Neural Networks II: Proceedings of the International Conference on Artificial Neural Networks*. Elsevier Publishers, 1992.

[6] K. Fukushima. Self-organisation of shift-invariant receptive fields. *Neural Networks*, 12:826–834, 1999.

[7] M. Stewart Bartlett and T.J. Sejnowski. Learning viewpoint invariant face representations from visual experience in an attractor network. *Network: Computation in Neural Systems*, 9(3):399–417, 1998.

[8] E. T. Rolls and T. Milward. A model of invariant object recognition in the visual system: Learning rules, activation functions, lateral inhibition, and information-based performance measures. *Neural Computation*, 12:2547–2572, 2000.

[9] J. V. Stone. Learning perceptually salient visual parameters using spatiotemporal smoothness constraints. *Neural Computation*, 8(7):1463–1492, October 1996.

[10] K. Kayser, W. Einhäuser, O. Dümmer, P. König, and K. Körding. Extracting slow subspaces from natural videos leads to complex cells. In *ICANN 2001, LNCS 2130*, pages 1075–1080. Springer-Verlag Berlin Heidelberg 2001, 2001.

[11] J. Hurri and A. Hyvarinen. Simple-cell-like receptive fields maximise temporal coherence in natural video. *Submitted, http://www.cis.hut.fi/~jarmo/publications*, 2002.

[12] D. Martinez and A. Bray. Nonlinear blind source separation using kernels. *IEEE Trans. Neural Networks*, 14(1):228–235, Jan. 2003.

[13] G. Baudat and F. Anouar. Kernel-based methods and function approximation. *International Joint Conference of Neural Networks IJCNN*, pages 1244–1249, 2001.

[14] A. J. Bell and T. J. Sejnowski. The independent components of natural scenes are edge filters. *Vision Research*, 37:3327–3338, 1997.

[15] B.A. Olhausen and D.J. Field. Emergence of simple-cell receptive field properties by learning a sparse code for natural images. *Nature*, 381:607–609, 1996.

[16] E.T. Rolls and G. Deco. *Computational Neuroscience of Vision*. Oxford University Press, 2002.
